# Recursive algorithms for approximating probabilities in graphical models

**Tommi S. Jaakkola and Michael I. Jordan**
{tommi,jordan}@psyche.mit.edu
Department of Brain and Cognitive Sciences
Massachusetts Institute of Technology
Cambridge, MA 02139

## Abstract

We develop a recursive node-elimination formalism for efficiently approximating large probabilistic networks. No constraints are set on the network topologies. Yet the formalism can be straightforwardly integrated with exact methods whenever they are/become applicable. The approximations we use are controlled: they maintain consistently upper and lower bounds on the desired quantities at all times. We show that Boltzmann machines, sigmoid belief networks, or any combination (i.e., chain graphs) can be handled within the same framework. The accuracy of the methods is verified experimentally.

## 1  Introduction

Graphical models (see, e.g., Lauritzen 1996) provide a medium for rigorously embedding domain knowledge into network models. The structure in these graphical models embodies the qualitative assumptions about the independence relationships in the domain while the probability model attached to the graph permits a consistent computation of belief (or uncertainty) about the values of the variables in the network. The feasibility of performing this computation determines the ability to make inferences or to learn on the basis of observations. The standard framework for carrying out this computation consists of exact probabilistic methods (Lauritzen 1996). Such methods are nevertheless restricted to fairly small or sparsely connected networks and the use of approximate techniques is likely to be the rule for highly interconnected graphs of the kind studied in the neural network literature.

There are several desiderata for methods that calculate approximations to posterior probabilities on graphs. Besides having to be (1) reasonably accurate and fast to compute, such techniques should yield (2) rigorous estimates of confidence about

the attained results; this is especially important in many real-world applications (e.g., in medicine). Furthermore, a considerable gain in accuracy could be obtained from (3) the ability to use the techniques in conjunction with exact calculations whenever feasible. These goals have been addressed in the literature with varying degrees of success. For inference and learning in Boltzmann machines, for example, classical mean field approximations (Peterson & Anderson, 1987) address only the first goal. In the case of sigmoid belief networks (Neal 1992), partial solutions have been provided to the first two goals (Dayan et al. 1995; Saul et al. 1996; Jaakkola & Jordan 1996). The goal of integrating approximations with exact techniques has been introduced in the context of Boltzmann machines (Saul & Jordan 1996) but nevertheless leaving the solution to our second goal unattained. In this paper, we develop a recursive node-elimination formalism that meets all three objectives for a powerful class of networks known as chain graphs (see, e.g., Lauritzen 1996); the chain graphs we consider are of a restricted type but they nevertheless include Boltzmann machines and sigmoid belief networks as special cases.

We start by deriving the recursive formalism for Boltzmann machines. The results are then generalized to sigmoid belief networks and the chain graphs.

## 2  Boltzmann machines

We begin by considering Boltzmann machines with binary (0/1) variables. We assume the joint probability distribution for the variables $S = \{S_1, \ldots, S_n\}$ to be given by

$$P_n(S|h, J) = \frac{1}{Z_n(h, J)} B_n(S|h, J) \tag{1}$$

where $h$ and $J$ are the vector of biases and weights respectively, and the Boltzmann factor $B$ has the form

$$B_n(S|h, J) = \exp\left\{\sum_{i=1}^{n} h_i S_i + \frac{1}{2} \sum_{i,j=1}^{n} J_{ij} S_i S_j\right\} \tag{2}$$

The partition function $Z_n(h, J) = \sum_S B_n(S|h, J)$ normalizes the distribution. The Boltzmann distribution defined in this manner is tractable insofar as we are able to compute the partition function; indeed, all marginal distributions can be reduced to ratios of partition functions in different settings.

We now turn to methods for computing the partition function. In special cases (e.g., trees or chains) the structure of the weight matrix $J_{ij}$ may allow us to employ exact methods for calculating $Z$. Although exact methods are not feasible in more generic networks, selective approximations may nevertheless restore their utility. The recursive framework we develop provides a general and straightforward methodology for combining approximate and exact techniques.

The crux of our approach lies in obtaining variational bounds that allow the creation of recursive node-elimination formulas of the form[1]:

$$Z_n(h, J) \lessgtr C(h, J) Z_{n-1}(\tilde{h}, \tilde{J}) \tag{3}$$

Such formulas are attractive for three main reasons: (1) a variable (or many at the same time) can be eliminated by merely transforming the model parameters ($h$ and $J$); (2) the approximations involved in the elimination are controlled, i.e., they

consistently yield upper or lower bounds at each stage of the recursion; (3) most importantly, if the remaining (simplified) partition function $Z_{n-1}(\tilde{h}, \tilde{J})$ allow the use of exact methods, the corresponding model parameters $\tilde{h}$ and $\tilde{J}$ can simply be passed on to such routines.

Next we will consider how to obtain the bounds and outline their implications. Note that since the quantities of interest are predominantly ratios of partition functions, it is the combination of upper and lower bounds that is necessary to rigorously bound the target quantities. This applies to parameter estimation as well even if only a lower bound on likelihood of examples is used; such likelihood bound relies on both upper and lower bounds on partition functions.

## 2.1  Simple recursive factorizations

We start by developing a lower bound recursion. Consider eliminating the variable $S_i$:

$$Z_n(h, J) \quad = \quad \sum_S B_n(S|h, J) = \sum_{S \backslash S_i} \sum_{S_i} B_n(S|h, J) \tag{4}$$

$$= \quad \sum_{S \backslash S_i} (1 + e^{h_i + \sum_j J_{ij} S_j}) B_{n-1}(S \backslash S_i|h, J) \tag{5}$$

$$\geq \quad \sum_{S \backslash S_i} e^{\mu_i(h_i + \sum_j J_{ij} S_j) + H(\mu_i)} B_{n-1}(S \backslash S_i|h, J) \tag{6}$$

$$= \quad e^{\mu_i h_i + H(\mu_i)} \sum_{S \backslash S_i} B_{n-1}(S \backslash S_i|\tilde{h}, J) \tag{7}$$

$$= \quad e^{\mu_i h_i + H(\mu_i)} Z_{n-1}(\tilde{h}, J) \tag{8}$$

where $\tilde{h}_j = h_j + \mu_i J_{ij}$ for $j \neq i$, $H(\cdot)$ is the binary entropy function and $\mu_i$ are free parameters that we will refer to as "variational parameters." The variational bound introduced in eq. (6) can be verified by a direct maximization which recovers the original expression. This lower bound recursion bears a connection to mean field approximation and in particular to the structured mean field approximation studied by Saul and Jordan (1996).[2]

Each recursive elimination translates into an additional bound and therefore the approximation (lower bound) deteriorates with the number of such iterations. It is necessary, however, to continue with the recursion only to the extent that the prevailing partition function remains unwieldy to exact methods. Consequently, the problem becomes that of finding the variables the elimination of which would render the rest of the graph tractable. Figure 1 illustrates this objective. Note that the simple recursion does not change the connection matrix $J$ for the remaining variables; thus, graphically, the operation translates into merely removing the variable.

The above recursive procedure maintains a lower bound on the partition function that results from the variational representation introduced in eq. (6). For rigorous

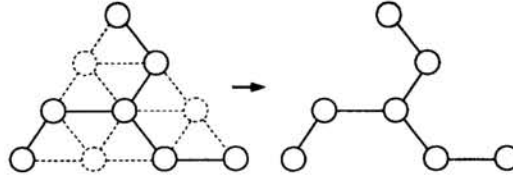

Figure 1: Enforcing tractable networks. Each variable in the graph can be removed (in any order) by adding the appropriate biases for the existing adjacent variables. The elimination of the dotted nodes reveals a simplified graph underneath.

bounds we need an upper bound as well. In order to preserve the graphical interpretation of the lower bound, the upper bound should also be factorized. With this in mind, the bound of eq. (6) can be replaced with

$$1 + e^{h_i + \sum_j J_{ij} S_j} \leq e^{\hat{h}_0 + \sum_j S_j \hat{h}_j} \tag{9}$$

where

$$\hat{h}_j = q_j \left[ f \left( J_{ij}/q_j + h_i \right) - f(h_i) \right] \tag{10}$$

for $j > 0$, $\hat{h}_0 = f(h_i)$, $f(x) = \log(1 + e^x)$, and $q_j$ are variational parameters such that $\sum_j q_j = 1$. The derivation of this bound can be found in appendix A.

## 2.2   Refined recursive bound

If the (sub)network is densely or fully connected the simple recursive methods presented earlier can hardly uncover any useful structure. Thus a large number of recursive steps are needed before relying on exact methods and the accuracy of the overall bound is compromised. To improve the accuracy, we introduce a more sophisticated variational (upper) bound to replace the one in eq. (6). By denoting $X_i = h_i + \sum_j J_{ij} S_j$ we have:

$$1 + e^{X_i} \leq e^{X_i/2 + \lambda(x_i) X_i^2 - F(\lambda, x_i)} \tag{11}$$

The derivation and the functional forms of $\lambda(x_i)$ and $F(\lambda, x_i)$ are presented in appendix B. We note here, however, that the bound is exact whenever $x_i = X_i$. In terms of the recursion we obtain

$$Z_n(h, J) \leq e^{h_i/2 + \lambda(x_i) h_i^2 - F(\lambda, x_i)} Z_{n-1}(\tilde{h}, \tilde{J}) \tag{12}$$

where

$$\tilde{h}_j = h_j + 2 h_i \lambda(x_i) J_{ij} + J_{ij}/2 + \lambda(x_i) J_{ij}^2 \tag{13}$$

$$\tilde{J}_{jk} = J_{jk} + 2 \lambda(x_i) J_{ji} J_{ik} \tag{14}$$

for $j \neq k \neq i$. Importantly and as shown in figure 2a, this refined recursion imposes (qualitatively) the proper structural changes on the remaining network: the variables adjacent to the eliminated (or marginalized) variable become connected. In other words, if $J_{ij} \neq 0$ and $J_{ik} \neq 0$ then $J_{jk} \neq 0$ after the recursion.

To substantiate the claim of improved accuracy we tested the refined upper bound recursion against the factorized lower bound recursion in random fully connected networks with 8 variables[3]. The weights in these networks were chosen uniformly in the range $[-d, d]$ and all the initial biases were set to zero. Figure 3a plots the relative errors in the log-partition function estimates for the two recursions as a

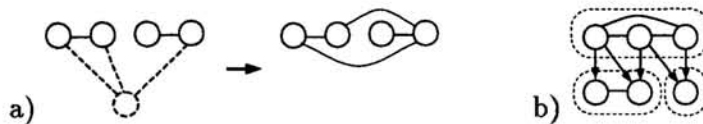

Figure 2: a) The graphical changes in the network following the refined recursion match those of proper marginalization. b) Example of a chain graph. The dotted ovals indicate the undirected clusters.

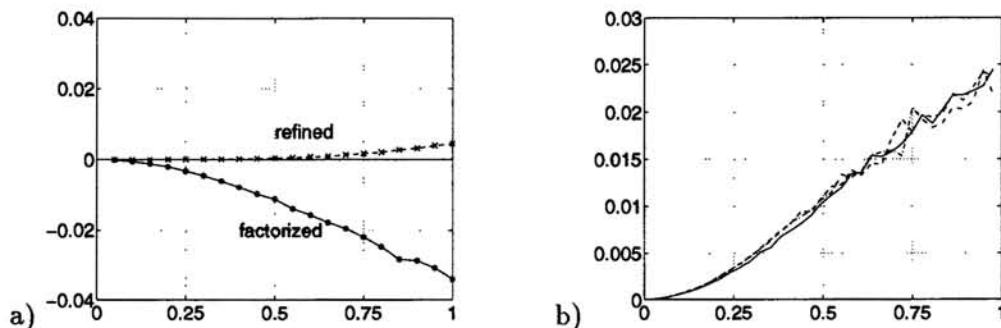

Figure 3: a) The mean relative errors in the log-partition function as a function of the scale of the random weights (uniform in $[-d, d]$). Solid line: factorized lower bound recursion; dashed line: refined upper bound. b) Mean relative difference between the upper and lower bound recursions as a function of $d\sqrt{n/8}$, where $n$ is the network size. Solid: $n = 8$; dashed: $n = 64$; dotdashed: $n = 128$.

function of the scale $d$. Figure 3b reveals how the relative difference between the two bounds is affected by the network size. In the illustrated scale the size has little effect on the difference. We note that the difference is mainly due to the factorized lower bound recursion as is evident from Figure 3a.

## 3 Chain graphs and sigmoid belief networks

The recursive bounds presented earlier can be carried over to chain graphs[4]. An example of a chain graph is given in figure 2b. The joint distribution for a chain graph can written as a product of conditional distributions for clusters of variables:

$$P_n(S|J) = \prod_k P(S^k|\text{pa}[k], h^k, J^k) \qquad (15)$$

where $S^k = \{S_i\}_{i \in C_k}$ is the set of variables in cluster $k$. In our case, the conditional probabilities for each cluster are conditional Boltzmann distributions given by

$$P(S^k|\text{pa}[k], h^k, J^k) = \frac{B(S^k|h_S^k, J^k)}{Z(h_S^k, J^k)} \qquad (16)$$

where the added complexity beyond that of ordinary Boltzmann machines is that the Boltzmann factors now include also outside cluster biases:

$$[h_S^k]_i = h_i^k + \sum_{j \notin C_k} J_{ij}^{k,out} S_j \qquad (17)$$

where the index $i$ stays within the $k^{th}$ cluster. We note that sigmoid belief networks correspond to the special case where there is only single binary variable in each cluster; Boltzmann machines, on the other hand, have only one cluster.

We now show that the recursive formalism can be extended to chain graphs. This is achieved by rewriting or bounding the conditional probabilities in terms of variational Boltzmann factors. Consequently, the joint distribution – being a product of the conditionals – will also be a Boltzmann factor. Computing likelihoods (marginals) from such a joint distribution amounts to calculating the value of a particular partition function and therefore reduces to the case considered earlier.

It suffices to find variational Boltzmann factors that bound (or rerepresent in some cases) the cluster partition functions in the conditional probabilities. We observe first that in the factorized lower bound or in the refined upper bound recursions, the initial biases will appear in the resulting expressions either linearly or quadratically in the exponent[5]. Since the initial biases for the clusters are of the form of eq. (17), the resulting expressions must be Boltzmann factors with respect to the variables outside the cluster. Thus, applying the recursive approximations to each cluster partition function yields an upper/lower bound in the form of a Boltzmann factor. Combining such bounds from each cluster finally gives upper/lower bounds for the joint distribution in terms of variational Boltzmann factors.

We note that for sigmoid belief networks the Boltzmann factors bounding the joint distribution are in fact exact variational translations of the true joint distribution. To see this, let us denote $X_i = \sum J_{ij} S_j + h_i$ and use the variational forms, for example, from eq. (6) and (11):

$$\sigma(X_i) = (1 + e^{-X_i})^{-1} \quad \leq \quad e^{\mu_i X_i - H(\mu_i)} \tag{18}$$

$$\geq \quad e^{X_i/2 - \lambda(x_i)X_i^2 + F(\lambda, x_i)} \tag{19}$$

where the sigmoid function $\sigma(\cdot)$ is the inverse cluster partition function in this case. Both the variational forms are Boltzmann factors (at most quadratic in $X_i$ in the exponent) and are exact if minimized/maximized with respect to the variational parameters.

In sum, we have shown how the joint distribution for chain graphs can be bounded by (translated into) Boltzmann factors to which the recursive approximation formalism is again applicable.

## 4   Conclusion

To reap the benefits of probabilistic formulations of network architectures, approximate methods are often unavoidable in real-world problems. We have developed a recursive node-elimination formalism for rigorously approximating intractable networks. The formalism applies to a large class of networks known as chain graphs and can be straightforwardly integrated with exact probabilistic calculations whenever they are applicable. Furthermore, the formalism provides rigorous upper and lower bounds on any desired quantity (e.g., the variable means) which is crucial in high risk application domains such as medicine.

## Footnotes

[1]Related schemes in the physics literature (renormalization group) are unsuitable here as they generally don't provide strict upper/lower bounds.

[2]Each lower bound recursion can be shown to be equivalent to a mean field approximation of the eliminated variable(s). The structured mean field approach of Saul and Jordan (1996) suggests using exact methods for tractable substructures while mean field for the variables mediating these structures. Translated into our framework this amounts to eliminating the mediating variables through the recursive lower bound formula with a subsequent appeal to exact methods. The connection is limited to the lower bound.

[3]The small network size was chosen to facilitate comparisons with exact results.

[4]While Boltzmann machines are undirected networks (interactions defined through potentials), sigmoid networks are directed models (constructed from conditional probabilities). Chain graphs contain both directed and undirected interactions.

[5]This follows from the linearity of the propagation rules for the biases, and the fact that the emerging prefactors are either linear or quadratic in the exponent.

## References

P. Dayan, G. Hinton, R. Neal, and R. Zemel (1995). The Helmholtz machine. *Neural Computation* **7**: 889-904.

S. L. Lauritzen (1996). *Graphical Models.* Oxford: Oxford University Press.

T. Jaakkola and M. Jordan (1996). Computing upper and lower bounds on likelihoods in intractable networks. To appear in *Proceedings of the twelfth Conference on Uncertainty in Artificial Intelligence.*

R. Neal. Connectionist learning of belief networks (1992). *Artificial Intelligence* **56**: 71-113.

C. Peterson and J. R. Anderson (1987). A mean field theory learning algorithm for neural networks. *Complex Systems* **1**: 995-1019.

L. K. Saul, T. Jaakkola, and M. I. Jordan (1996). Mean field theory for sigmoid belief networks. *JAIR* **4**: 61-76.

L. Saul and M. Jordan (1996). Exploiting tractable substructures in intractable networks. To appear in *Advances of Neural Information Processing Systems 8. MIT Press.*

## A    Factorized upper bound

The bound follows from the convexity of $f(x) = \log(1+e^x)$ and from an application of Jensen's inequality. Let $f_k(x) = f(x + h_k)$ and note that $f_k(x)$ has the same convexity properties as $f$. For any convex function $f_k$ then we have (by Jensen's inequality)

$$f_k\left(\sum_j J_{kj}S_j\right) = f_k\left(\sum_j q_j \frac{J_{kj}S_j}{q_j}\right) \leq \sum_j q_j f_k\left(\frac{J_{kj}S_j}{q_j}\right) \tag{20}$$

By rewriting $f_k\left(\frac{J_{kj}S_j}{q_j}\right) = S_j\left[f_k\left(\frac{J_{kj}}{q_j}\right) - f_k(0)\right] + f_k(0)$ we get the desired result.

## B    Refined upper bound

To derive the upper bound consider first

$$1 + e^x = e^{x/2} + \log(e^{-x/2} + e^{x/2}) \tag{21}$$

Now, $g(x) = \log(e^{-x/2} + e^{x/2})$ is a symmetric function of $x$ and also a concave function of $x^2$. Any tangent line for a concave function always remains above the function and so it also serves as an upper bound. Therefore we may bound $g(x)$ by the tangents of $g(\sqrt{y})$ (due to the concavity in $x^2$). Thus

$$\log(e^{-x/2} + e^{x/2}) \leq \frac{\partial g(\sqrt{y})}{\partial y}(x^2 - y) + g(\sqrt{y}) \tag{22}$$

$$= \lambda(y)x^2 - F(\lambda, y) \tag{23}$$

where

$$\lambda(y) = \frac{\partial}{\partial y}g(\sqrt{y}) \tag{24}$$

$$F(\lambda, y) = \lambda(y)\, y - g(\sqrt{y}) \tag{25}$$

The desired result now follows the change of variables: $y = x_i^2$. Note that the tangent bound is exact whenever $x_i = x$ (a tangent defined at that point).